# The Perceptron Algorithm Is Fast for Non-Malicious Distributions

**Erice B. Baum**
NEC Research Institute
4 Independence Way
Princeton, NJ 08540

**Abstract:** Within the context of Valiant's protocol for learning, the Perceptron algorithm is shown to learn an arbitrary half-space in time $O(\frac{n^2}{\epsilon^3})$ if $D$, the probability distribution of examples, is taken uniform over the unit sphere $S^n$. Here $\epsilon$ is the accuracy parameter. This is surprisingly fast, as "standard" approaches involve solution of a linear programming problem involving $\Omega(\frac{n}{\epsilon})$ constraints in $n$ dimensions. A modification of Valiant's distribution independent protocol for learning is proposed in which the distribution and the function to be learned may be chosen by adversaries, however these adversaries may not communicate. It is argued that this definition is more reasonable and applicable to real world learning than Valiant's. Under this definition, the Perceptron algorithm is shown to be a distribution independent learning algorithm. In an appendix we show that, for uniform distributions, some classes of infinite V-C dimension including convex sets and a class of nested differences of convex sets are learnable.

## §1: Introduction

The Perceptron algorithm was proved in the early 1960s[Rosenblatt,1962] to converge and yield a half space separating any set of linearly separable classified examples. Interest in this algorithm waned in the 1970's after it was emphasized[Minsky and Papert, 1969] (1) that the class of problems solvable by a single half space was limited, and (2) that the Perceptron algorithm, although converging in finite time, did not converge in polynomial time. In the 1980's, however, it has become evident that there is no hope of providing a learning algorithm which can learn arbitrary functions in polynomial time and much research has thus been restricted to algorithms which learn a function drawn from a particular class of functions. Moreover, learning theory has focused on protocols like that of [Valiant, 1984] where we seek to classify, not a fixed set of examples, but examples drawn from a probability distribution. This allows a natural notion of "generalization". There are very few classes which have yet been proven learnable in polynomial time, and one of these is the class of half spaces. Thus there is considerable theoretical interest now in studying the problem of learning a single half space, and so it is natural to reexamine the Perceptron algorithm within the formalism of Valiant.

In Valiant's protocol, a class of functions is called learnable if there is a learning algorithm which works in polynomial time independent of the distribution $D$ generating the examples. Under this definition the Perceptron learning algorithm is not a polynomial time learning algorithm. However we will argue in section 2 that this definition is too restrictive. We will consider in section 3 the behavior of the Perceptron algorithm if $D$ is taken to be the uniform distribution on the unit sphere $S^n$. In this case, we will see that the Perceptron algorithm converges remarkably rapidly. Indeed we will give a time bound which is faster than any bound known to us for any algorithm solving this problem. Then, in section 4, we will present what we believe to be a more natural definition of distribution independent learning in this context, which we will call Nonmalicious distribution independent learning. We will see that the Perceptron algorithm is indeed a polynomial time non-malicious distribution independent learning algorithm. In Appendix A, we sketch proofs that, if one restricts attention to the uniform distribution, some classes with infinite Vapnik-Chervonenkis dimension such as the class of convex sets and the class of nested differences of convex sets (which we define) are learnable. These results support our assertion that distribution independence is too much to ask for, and may also be of independent interest.

## §2: Distribution Independent Learning

In Valiant's protocol[Valiant, 1984], a class $F$ of Boolean functions on $\Re^n$ is called learnable if a learning algorithm $A$ exists which satisfies the following conditions. Pick some probability distribution $D$ on $\Re^n$. $A$ is allowed to call examples, which are pairs $(x, f(x))$, where $x$ is drawn according to the distribution $D$. $A$ is a valid learning algorithm for $F$ if for any probability distribution $D$ on $\Re^n$, for any $0 < \delta, \epsilon < 1$, for any $f \in F$, $A$ calls examples and, with probability at least $1 - \delta$ outputs in time bounded by a polynomial in $n, \delta^{-1}$, and $\epsilon^{-1}$ a hypothesis $g$ such that the probability that $f(x) \neq g(x)$ is less than $\epsilon$ for $x$ drawn according to $D$.

This protocol includes a natural formalization of 'generalization' as prediction.For more discussion see [Valiant, 1984]. The definition is restrictive in demanding that $A$ work for an arbitrary probability distribution $D$. This demand is suggested by results on uniform convergence of the empirical distribution to the actual distribution. In particular, if $F$ has Vapnik-Chervonenkis (V-C) dimension[1] $d$, then it has been proved[Blumer et al, 1987] that all $A$ needs to do to be a valid learning algorithm is to call $M_0(\epsilon, \delta, d) = max(\frac{4}{\epsilon} log \frac{2}{\delta}, \frac{8d}{\epsilon} log \frac{13}{\epsilon})$ examples and to find in polynomial time a function $g \in F$ which correctly classifies these.

Thus, for example, it is simple to show that the class $H$ of half spaces is Valiant learnable[Blumer et al, 1987]. The V-C dimension of $H$ is $n + 1$. All we need to do to learn $H$ is to call $M_0(\epsilon, \delta, n + 1)$ examples and find a separating half space using Karmarkar's algorithm [Karmarkar, 1984]. Note that the Perceptron algorithm would not work here, since one can readily find distributions for which the Perceptron algorithm would be expected to take arbitrarily long times to find a separating half space.

Now, however, it seems from three points of view that the distribution independent definition is too strong. First, although the results of [Blumer et al., 1987] tell us we can gather enough information for learning in polynomial time, they say nothing about when we can actually find an algorithm A which learns in polynomial time. So far, such algorithms have only been found in a few cases, and (see, e.g. [Baum, 1989a]) these cases may be argued to be trivial.

Second, a few classes of functions have been proved (modulo strong but plausible complexity theoretic hypotheses) unlearnable by construction of cryptographically secure subclasses. Thus for example [Kearns and Valiant, 1988] show that the class of feedforward networks of threshold gates of some constant depth, or of Boolean gates of logarithmic depth, is not learnable by construction of a cryptographically secure subclass. The relevance of such results to learning in the natural world is unclear to us. For example, these results do not rule out a learning algorithm that would learn almost any log depth net. We would thus prefer a less restrictive definition of learnability, so that if a class were proved unlearnable, it would provide a meaningful limit on pragmatic learning.

Third, the results of [Blumer et al, 1987] imply that we can only expect to learn a class of functions $F$ if $F$ has finite V-C dimension. Thus we are in the position of assuming an enormous amount of information about the class of functions to be learned- namely that it be some specific class of finite V-C dimension, but nothing whatever about the distribution of examples. In the real world, by contrast, we are likely to know at least as much about the distribution $D$ as we know about the class of functions $F$. If we relax the distribution independence criterion, then it can be shown that classes of infinite Vapnik-Chervonenkis dimension are learnable. For example, for the uniform distribution, the class of convex sets and a class of nested differences of convex sets ( both of which trivially have infinite V-C dimension) are shown to be learnable in Appendix A.

## §3: The Perceptron Algorithm and Uniform Distributions

The Perceptron algorithm yields, in finite time, a half-space $(w_H, \theta_H)$ which correctly classifies any given set of linearly separable examples [Rosenblatt,1962]. That is, given a set of classified examples $\{x_\pm^\mu\}$ such that, for some $(w_t, \theta_t)$, $w_t \cdot x_+^\mu > \theta_t$ and $w_t \cdot x_-^\mu < \theta_t$ for all $\mu$, the algorithm converges in finite time to output a $(w_H, \theta_H)$ such that $w_H \cdot x_+^\mu \geq \theta_H$ and $w_H \cdot x_-^\mu < \theta_H$. We will normalize so that $\vec{w}_t \cdot \vec{w}_t = 1$. Note that $|w_t \cdot x - \theta_t|$ is the Euclidean distance from $x$ to the separating hyperplane $\{y : w_t \cdot y = \theta_t\}$.

The algorithm is the following. Start with some initial candidate $(w_0, \theta_0)$, which we will take to be $(\vec{0}, 0)$. Cycle through the examples. For each example, test whether that example is correctly classified. If so, proceed to the next example. If not, modify the candidate by

$$(w_{k+1} = w_k \pm x_\pm^\mu, \theta_{k+1} = \theta_k \mp 1) \tag{1}$$

where the sign of the modification is determined by the classification of the missclassified example.

In this section we will apply the Perceptron algorithm to the problem of learning

in the probabilistic context described in section 2, where however the distribution $D$ generating examples is uniform on the unit sphere $S^n$. Rather than have a fixed set of examples, we apply the algorithm in a slightly novel way: we call an example, perform a Perceptron update step, discard the example, and iterate until we converge to accuracy $\epsilon$.[f2] If we applied the Perceptron algorithm in the standard way, it seemingly would not converge as rapidly. We will return to this point at the end of this section.

Now the number of updates the Perceptron algorithm must make to learn a given set of examples is well known to be $O(\frac{1}{l^2})$, where $l$ is the minimum distance from an example to the classifying hyperplane (see eg. [Minsky and Papert, 1969]). In order to learn to $\epsilon$ accuracy in the sense of Valiant, we will observe that for the uniform distribution we do not need to correctly classify examples closer to the target separating hyperplane than $\Omega(\frac{\epsilon}{\sqrt{n}})$. Thus we will prove that the Perceptron algorithm will converge (with probability $1 - \delta$) after $O(\frac{n}{\epsilon^2})$ updates, which will occur after $O(\frac{n}{\epsilon^3})$ presentations of examples.

Indeed take $\theta_t = 0$ so the target hyperplane passes through the origin. Parallel hyperplanes a distance $\kappa/2$ above and below the target hyperplane bound a band B of probability measure

$$P(\kappa) = \int_{-\kappa/2}^{\kappa/2} \left(\sqrt{1 - z^2}\right)^{n-2} dz \, \frac{A_{n-1}}{A_n} \qquad (2)$$

(for $n \geq 2$), where $A_n = \frac{2\pi^{(n+1)/2}}{\Gamma((n+1)/2)}$ is the area of $S^n$. See figure 1. Using the readily

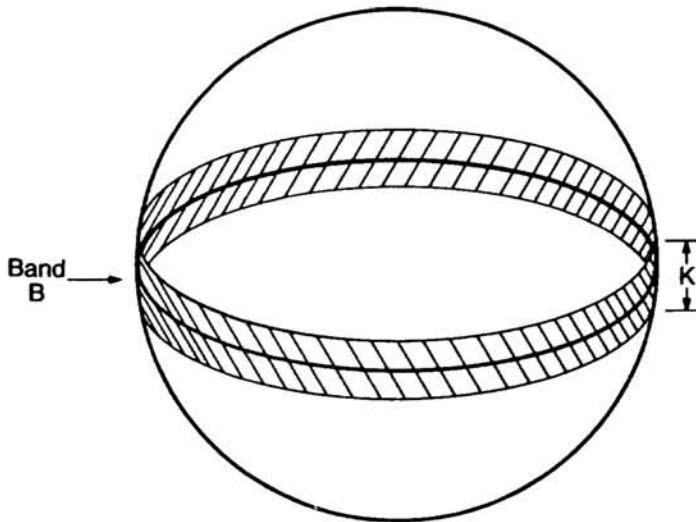

**Figure 1:** The target hyperplane intersects the sphere $S^n$ along its equator (if $\theta_t = 0$) shown as the central line. Points in (say) the upper hemisphere are classified as positive examples and those in the lower as negative examples. The band $B$ is formed by intersecting the sphere with two planes parallel to the target hyperplane and a distance $\kappa/2$ above and below it.

---

[f2]   We say that our candidate half space has accuracy $\epsilon$ when the probability that it missclassifies an example drawn from $D$ is no greater than $\epsilon$.

obtainable (e.g. by Stirling's formula) bound that $\frac{A_{n-1}}{A_n} < \sqrt{n}$, and the fact that the integrand is nowhere greater than 1, we find that for $\kappa = \epsilon/2\sqrt{n}$, the band has measure less than $\epsilon/2$. If $\theta_t \neq 0$, a band of width $\kappa$ will have less measure than it would for $\theta_t = 0$. We will thus continue to argue (without loss of generality) by assuming the worst case condition that $\theta_t = 0$.

Since B has measure less than $\epsilon/2$, if we have not yet converged to accuracy $\epsilon$, there is no more than probability $1/2$ that the next example on which we update will be in $B$. We will show that once we have made $m_0 = max(144ln\frac{\delta}{2}, \frac{48}{\kappa^2})$ updates, we have converged unless more than $7/12$ of the updates are in $B$. The probability of making this fraction of the updates in $B$, however, is less than $\delta/2$ if the probability of each update lying in $B$ is not more than $1/2$. We conclude with confidence $1-\delta/2$ that the probability our next update will be in $B$ is greater than $1/2$ and thus that we have converged to $\epsilon$-accuracy.

Indeed, consider the change in the quantity

$$N(\alpha) = \| \alpha w_t - w_k \|^2 + \| \alpha\theta_t - \theta_k \|^2 \tag{3}$$

when we update.

$$\Delta N \equiv \| \alpha w_t - w_{k+1} \|^2 + \| \alpha\theta_t - \theta_{k+1} \|^2 - \| \alpha w_t - w_k \|^2 - \| \alpha\theta_t - \theta_k \|^2 =$$

$$\mp 2\alpha w_t \cdot x_\pm \pm 2\alpha\theta_t \pm 2w_k \cdot x_\pm \mp 2\theta_k + \| x \|^2 + 1. \tag{4}$$

Now note that $\pm(w_k \cdot x_\pm - \theta_k) < 0$ since $x$ was missclassified by $(w_k, \theta_k)$ (else we would not update). Let $A = (\mp(w_t \cdot x_\pm - \theta_t))$. If $x \in B$, then $A \leq 0$. If $x \notin B$, then $A \leq -\kappa/2$. Recalling $x^2 = 1$, we see that $\Delta N < 2$ for $x \in B$ and $\Delta N < -\alpha\kappa + 2$ for $x \notin B$. If we choose $\alpha = 8/\kappa$, we find that $\Delta N \leq -6$ for $x \notin B$. Recall that, for $k = 0$, with $(w_0, \theta_0) = (0, 0)$, we have $N = \alpha^2 = 64/\kappa^2$. Thus we see that if we have made $O$ updates on points outside $B$, and $I$ updates on points in $B$, $N < 0$ if $6O - 2I > 64/\kappa^2$. But $N$ is positive semidefinite. Once we have made $48/\kappa^2$ total updates, at least $7/12$ of the updates must thus have been on examples in $B$.

If you assume that the probability of updates falling in $B$ is less than $1/2$ (and thus that our hypothesis half space is not yet at $\epsilon$ - accuracy), then the probability that more than $7/12$ of $m_0 = max(144ln\frac{\delta}{2}, \frac{48}{\kappa^2})$ updates fall in $B$ is less than $\delta/2$. To see this define $LE(p, m, r)$ as the probability of having at most $r$ successes in $m$ independent Bernoulli trials with probability of success $p$ and recall, [Angluin and Valiant,1979], for $0 \leq \beta \leq 1$ that

$$LE(p, m, (1-\beta)mp) \leq e^{-\beta^2 mp/2}. \tag{5}$$

Applying this formula with $m = m_0, p = 1/2, \beta = 1/6$ shows the desired result. We conclude that the probability of making $m_0$ updates without converging to $\epsilon$ accuracy is less than $\delta/2$.

However, as it approaches $1 - \epsilon$ accuracy, the algorithm will only update on a fraction $\epsilon$ of the examples. To get, with confidence $1 - \delta/2$, $m_0$ *updates*, it suffices to call $M = 2m_o/\epsilon$ examples. Thus we see that the Perceptron algorithm converges, with confidence $1 - \delta$, after we have called

$$M = \frac{2}{\epsilon}max(144ln\frac{\delta}{2}, \ \frac{48n}{\epsilon^2}) \tag{6}$$

examples.

Each example could be processed in time of order 1 on a "neuron" which computes $w_k \cdot x$ in time 1 and updates each of its "synaptic weights" in parallel. On a serial computer, however, processing each example will take time of order $n$, so that we have a time of order $O(n^2/\epsilon^3)$ for convergence on a serial computer.

This is remarkably fast. The general learning procedure, described in section 2, is to call $M_0(\epsilon, \delta, n+1)$ examples and find a separating halfspace, by some polynomial time algorithm for linear programming such as Karmarkar's algorithm. This linear programming problem thus contains $\Omega(\frac{n}{\epsilon})$ constraints in $n$ dimensions. Even to write down the problem thus takes time $\Omega(\frac{n^2}{\epsilon})$. The upper time bound to solve this given by [Karmarkar, 1984] is $O(n^{5.5}\epsilon^{-2})$. For large $n$ the Perceptron algorithm is faster by a factor of $n^{3.5}$. Of course it is likely that Karmarkar's algorithm could be proved to work faster than $\Omega(n^{5.5})$ for the particular distribution of examples of interest. If, however, Karmarkar's algorithm requires a number of iterations depending even logarithmically on $n$, it will scale worse (for large $n$) than the Perceptron algorithm.[3]

Notice also that if we simply called $M_0(\epsilon, \delta, n + 1)$ examples and used the Perceptron algorithm, in the traditional way, to find a linear separator for this set of examples, our time performance would not be nearly as good. In fact, equation 2 tells us that we would expect one of these examples to be a distance $O(\frac{\epsilon}{n^{1.5}})$ from the target hyperplane, since we are calling $\Omega(\frac{n}{\epsilon})$ examples and a band of width $O(\frac{\epsilon}{n^{1.5}})$ has measure $\Omega(\frac{\epsilon}{n})$. Thus this approach would take time $\Omega(\frac{n^4}{\epsilon^3})$, or a factor of $n^2$ worse than the one we have proposed.

An alternative approach to learning using only $O(\frac{n}{\epsilon})$ examples, would be to call $M_0(\frac{\epsilon}{4}, \delta, n + 1)$ examples and apply the Perceptron algorithm to these until a fraction $1 - \epsilon/2$ had been correctly classified. This would suffice to assure that the hypothesis half space so generated would (with confidence $1 - \delta$) have error less than $\epsilon$, as is seen from [Blumer et al, 1987, Theorem A3.3]. It is unclear to us what time performance this procedure would yield.

## §4: Non-Malicious Distribution Independent Learning

Next we propose modification of the distribution independence assumption, which we have argued is too strong to apply to real world learning. We begin with an informal description. We allow an adversary (adversary 1) to choose the

function $f$ in the class $F$ to present to the learning algorithm $A$. We allow a second adversary (adversary 2) to choose the distribution $D$ arbitrarily. We demand that (with probability $1 - \delta$) $A$ converge to produce an $\epsilon$-accurate hypothesis $g$. Thus far we have not changed Valiant's definition. Our restriction is simply that before their choice of distribution and function, adversaries 1 and 2 are not allowed to exchange information. Thus they must work independently. This seems to us an entirely natural and reasonable restriction in the real world.

Now if we pick any distribution and any hyperplane *independently*, it is highly unlikely that the probability measure will be concentrated close to the hyperplane. Thus we expect to see that under our restriction, the Perceptron algorithm is a distribution independent learning algorithm for $H$ and converges in time $O(\frac{n^2}{\epsilon^3 \delta^2})$ on a serial computer.

If adversary 1 and adversary 2 do not exchange information, the least we can expect is that they have no notion of a preferred direction on the sphere. Thus our informal demand that these two adversaries do not exchange information should imply, at least, that adversary 1 is equally likely to choose any $w_t$ (relative e.g. to whatever direction adversary 2 takes as his $z$ axis). This formalizes, sufficiently for our current purposes, the notion of Nonmalicious Distribution Independence.

**Theorem 1:** Let U be the uniform probability measure on $S^n$ and $D$ any other probability distribution on $S^n$. Let $R$ be any region on $S^n$ of U-measure $\epsilon \delta$ and let $x$ label some point in $R$. Choose a point $y$ on $S^n$ randomly according to $U$. Consider the region $R'$ formed by translating $R$ rigidly so that $x$ is mapped to $y$. Then the probability that the measure $D(R') > \epsilon$ is less than $\delta$.

*Proof:* Fix any point $z \in S^n$. Now choose $y$ and thus $R'$. The probability $z \in R'$ is $\epsilon \delta$. Thus in particular, if we choose a point $p$ according to $D$ and then choose $R'$, the probability that $p \in R'$ is $\epsilon \delta$.

Now assume that there is probability greater than $\delta$ that $D(R') > \epsilon$. Then we arrive immediately at a contradiction, since we discover that the probability that $p \in R'$ is greater than $\epsilon \delta$. **Q.E.D.**

**Corollary 2:** The Perceptron algorithm is a Non-malicious distribution independent learning algorithm for half spaces on the unit sphere which converges, with confidence $1 - \delta$ to accuracy $1 - \epsilon$ in time of order $O(\frac{n^2}{\epsilon^3 \delta^2})$ on a serial computer.

*Proof sketch:* Let $\kappa' = \epsilon \delta / 2\sqrt{n}$. Apply Theorem 1 to show that a band formed by hyperplanes a distance $\kappa'/2$ on either side of the target hyperplane has probability less than $\delta$ of having measure for examples greater than $\epsilon/2$. Then apply the arguments of the last section, with $\kappa'$ in place of $\kappa$. **Q.E.D.**

## Appendix A: Convex Sets Are Learnable for Uniform Distribution

In this appendix we sketch proofs that two classes of functions with infinite V-C dimension are learnable. These classes are the class of convex sets and a class of nested differences of convex sets which we define. These results support our

conjecture that full distribution independence is too restrictive a criterion to ask for if we want our results to have interesting applications. We believe these results are also of independent interest.

**Theorem 3:** The class $C$ of convex sets is learnable in time polynomial in $\epsilon^{-1}$ and $\delta^{-1}$ if the distribution of examples is uniform on the unit square in $d$ dimensions.

**Remarks:** (1) $C$ is well known to have infinite V-C dimension. (2) So far as we know, $C$ is not learnable in time polynomial in $d$ as well.

*Proof Sketch:*[14] We work, for simplicity, in 2 dimensions. Our arguments can readily be extended to $d$ dimensions.

The learning algorithm is to call $M$ examples (where $M$ will be specified). The positive examples are by definition within the convex set to be learned. Let $M_+$ be the set of positive examples. We classify examples as negative if they are linearly separable from $M_+$, i.e. outside of $c_+$, the convex hull of $M_+$.

Clearly this approach will never missclassify a negative example, but may missclassify positive examples which are outside $c_+$ and inside $c_t$. To show $\epsilon$- accuracy,

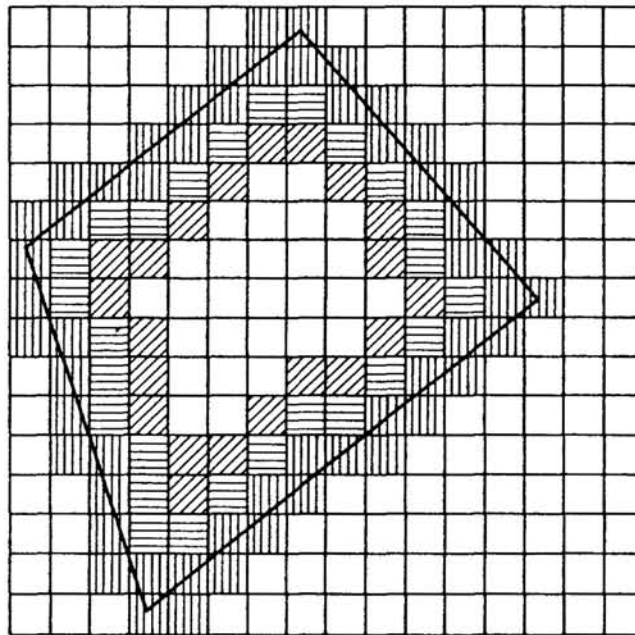

**Figure 2:** The boundary of the target concept $c_t$ is shown. The set $I_1$ of little squares intersecting the boundary of $c_t$ are hatched vertically. The set $I_2$ of squares just inside $I_1$ are hatched horizontally. The set $I_3$ of squares just inside $I_2$ are hatched diagonally. If we have an example in each square in $I_2$, the convex hull of these examples contains all points inside $c_t$ except possibly those in $I_1, I_2,$ or $I_3$.

we must choose $M$ large enough so that, with confidence $1 - \delta$, the symmetric difference of the target set $c_t$ and $c_+$ has area less than $\epsilon$.

Divide the unit square into $k^2$ equal subsquares. (See figure 2.) Call the set of subsquares which the boundary of $c_t$ intersects $I_1$. It is easy to see that the cardinality of $I_1$ is no greater than $4k$. The set $I_2$ of subsquares just inside $I_1$ also has cardinality no greater than $4k$, and likewise for the set $I_3$ of subsquares just inside $I_2$. If we have an example in each of the squares in $I_2$, then $c_t$ and $c_+$ clearly have symmetric difference at most equal the area of $I_1 \cup I_2 \cup I_3 \leq 12k \times k^{-2} = 12/k$. Thus take $k = 12/\epsilon$. Now choose $M$ sufficiently large so that after $M$ trials there is less than $\delta$ probability we have not got an example in each of the $4k$ squares in $I_2$. Thus we need $LE(k^{-2}, M, 4k) < \delta$. Using equation 5, we see that $M = \frac{500}{\epsilon^2} ln\delta$ will suffice. **Q.E.D.**

Actually, one can learn (for uniform distributions) a more complex class of functions formed out of nested convex regions. For any set $\{c_1, c_2, ..., c_l\}$ of $l$ convex regions in $\Re^d$, let $R_1 = c_1$ and for $j = 2, ..., l$ let $R_j = R_{j-1} \cap c_j$. Then define a concept $f = R_1 - R_2 + R_3 - ...R_l$. The class $C$ of concepts so formed we call nested convex sets. See figure 3.

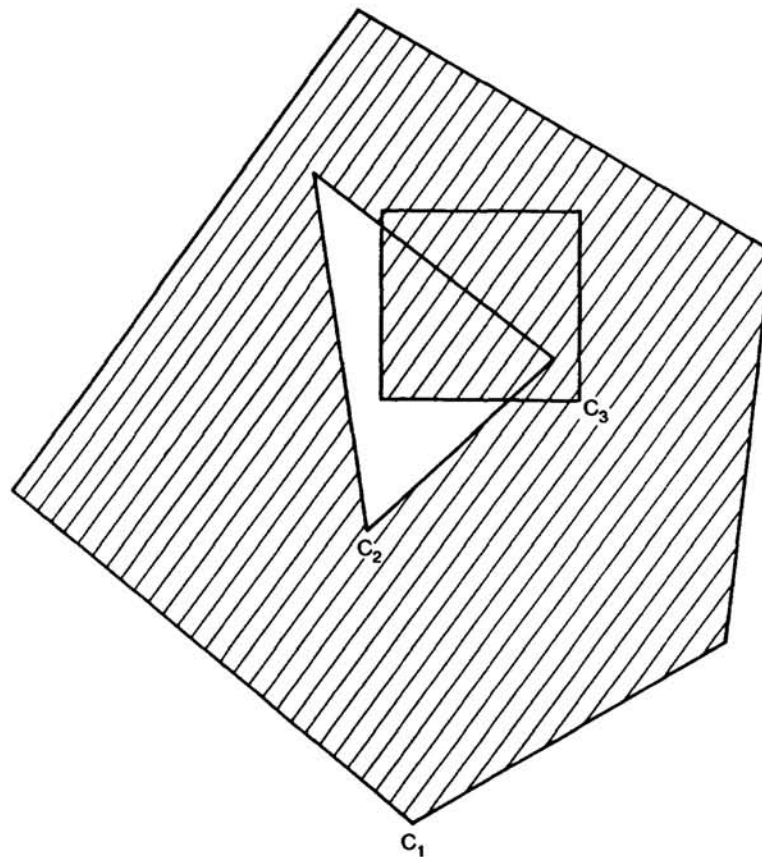

**Figure 3:** $c_1$ is the five sided region, $c_2$ is the triangular region, and $c_3$ is the square. The positive region $c_1 - c_2 \cup c_1 + c_3 \cup c_2 \cup c_1$ is shaded.

This class can be learned by an iterative procedure which peels the onion. Call a sufficient number of examples. (One can easily see that a number polynomial in $l, \epsilon$, and $\delta$ but of course exponential in $d$ will suffice.) Let the set of examples so obtained be called $S$. Those negative examples which are linearly separable from all positive examples are in the outermost layer. Class these in set $S_1$. Those positive examples which are linearly separable from all negative examples in $S - S_1$ lie in the next layer- call this set of positive examples $S_2$. Those negative examples in $S - S_1$ linearly separable from all positive examples in $S - S_2$ lie in the next layer, $S_3$. In this way one builds up $l + 1$ sets of examples. (Some of these sets may be empty.) One can then apply the methods of Theorem 3 to build a classifying function from the outside in. If the innermost layer $S_{l+1}$ is (say) negative examples, then any future example is called negative if it is not linearly separable from $S_{l+1}$, or is linearly separable from $S_l$ and not linearly separable from $S_{l-1}$, or is linearly separable from $S_{l-2}$ but not linearly separable from $S_{l-3}$, etc.

*Acknowledgement:* I would like to thank L.E. Baum for conversations and L. G. Valiant for comments on a draft. Portions of the work reported here were performed while the author was an employee of Princeton University and of the Jet Propulsion Laboratory, California Institute of Technology, and were supported by NSF grant DMR-8518163 and agencies of the US Department of Defence including the Innovative Science and Technology Office of the Strategic Defence Initiative Organization.

## Footnotes

[1] We say a set $S \subset R^n$ is shattered by a class $F$ of Boolean functions if $F$ induces all Boolean functions on $S$. The V-C dimension of $F$ is the cardinality of the largest set $S$ which $F$ shatters.

[3] We thank P. Vaidya for a discussion on this point.

[14] This proof is inspired by arguments presented in [Pollard, 1984], pp22-24. After this proof was completed, the author heard D. Haussler present related, unpublished results at the 1989 Snowbird meeting on Neural Computation.

## References

ANGLUIN, D., VALIANT, L.G. (1979), Fast probabilistic algorithms for Hamiltonian circuits and matchings, J. of Computer and Systems Sciences, 18, pp 155-193.

BAUM, E.B., (1989), On learning a union of half spaces, Journal of Complexity V5, N4.

BLUMER, A., EHRENFEUCHT,A., HAUSSLER,D., and WARMUTH,M. (1987), Learnability and the Vapnik-Chervonenkis Dimension, U.C.S.C. tech. rep. UCSC-CRL-87-20, and J. ACM, to appear.

KARMARKAR, N., (1984), A new polynomial time algorithm for linear programming, Combinatorica 4, pp373-395

KEARNS, M, and VALIANT, L., (1989), Cryptographic limitations on learning Boolean formulae and finite automata, Proc. 21st ACM Symp. on Theory of Computing, pp433-444.

MINSKY, M, and PAPERT,S., (1969), *Perceptrons, and Introduction to Computational Geometry*, MIT Press, Cambridge MA.

POLLARD, D. (1984), *Convergence of stochastic processes*, New York: Springer-Verlag.

ROSENBLATT, F. (1962), *Principles of Neurodynamics*, Spartan Books, N.Y.

VALIANT, L.G., (1984), A theory of the learnable, Comm. of ACM V27, N11, pp1134-1142.